# Efficient Principled Learning of Thin Junction Trees

**Anton Chechetka    Carlos Guestrin**
Carnegie Mellon University

## Abstract

We present the first truly polynomial algorithm for PAC-learning the structure of bounded-treewidth junction trees – an attractive subclass of probabilistic graphical models that permits both the compact representation of probability distributions and efficient exact inference. For a constant treewidth, our algorithm has polynomial time and sample complexity. If a junction tree with sufficiently strong intra-clique dependencies exists, we provide strong theoretical guarantees in terms of $KL$ divergence of the result from the true distribution. We also present a lazy extension of our approach that leads to very significant speed ups in practice, and demonstrate the viability of our method empirically, on several real world datasets.

One of our key new theoretical insights is a method for bounding the conditional mutual information of arbitrarily large sets of variables with only polynomially many mutual information computations on fixed-size subsets of variables, if the underlying distribution can be approximated by a bounded-treewidth junction tree.

## 1  Introduction

In many applications, e.g., medical diagnosis or datacenter performance monitoring, **probabilistic inference** plays an important role: to decide on a patient's treatment, it is useful to know the probability of various illnesses given the known symptoms. Thus, it is important to be able to represent probability distributions compactly and perform inference efficiently. Here, probabilistic graphical models (PGMs) have been successful as compact representations for probability distributions.

In order to use a PGM, one needs to define its structure and parameter values. Usually, we only have data (i.e., samples from a probability distribution), and learning the structure from data is thus a crucial task. For most formulations, the structure learning problem is NP-complete, *c.f.*, [10]. Most structure learning algorithms only guarantee that their output is a local optimum. One of the few notable exceptions is the work of Abbeel et al. [1], for learning structure of factor graphs, that provides probably approximately correct (PAC) learnability guarantees.

While PGMs can represent probability distributions compactly, exact inference in compact models, such as those of Abbeel et al., remains intractable [7]. An attractive solution is to use junction trees (JTs) of limited treewidth – a subclass of PGMs that permits efficient exact inference. For treewidth $k = 1$ (trees), the most likely (MLE) structure of a junction tree can be learned efficiently using the Chow-Liu algorithm [6], but the representational power of trees is often insufficient. We address the problem of learning JTs for fixed treewidth $k > 1$. Learning the most likely such JT is NP-complete [10]. While there are algorithms with global guarantees for learning fixed-treewidth JTs [10, 13], there has been no polynomial algorithm with PAC guarantees. The guarantee of [10] is in terms of the difference in log-likelihood of the MLE JT and the model where all variables are independent: the result is guaranteed to achieve at least a constant fraction of that difference. The constant does not improve as the amount of data increases, so it does not imply PAC learnability. The algorithm of [13] has PAC guarantees, but its complexity is exponential. In contrast, we provide a truly polynomial algorithm with PAC guarantees. The contributions of this paper are as follows:

- A theoretical result (Lemma 4) that upper bounds the conditional mutual information of arbitrarily large sets of random variables in polynomial time. In particular, we do not assume that an efficiently computable mutual information oracle exists.
- The first polynomial algorithm for PAC-learning the structure of limited-treewidth junction trees with strong intra-clique dependencies. We provide graceful degradation guarantees for distributions that are only approximately representable by JTs with fixed treewidth.

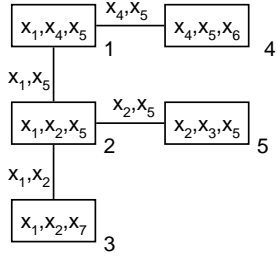

Figure 1: A junction tree. Rectangles denote cliques, separators are marked on the edges.

- A lazy heuristics that allows to make the algorithm practical.
- Empirical evidence of the viability of our approach on real-world datasets.

## 2   Bounded treewidth graphical models

In general, even to represent a probability distribution $P(V)$ over discrete variables[1] $V$ we need space exponential in the size $n$ of $V$. However, junction trees of limited treewidth allow compact representation *and* tractable exact inference. We briefly review junction trees (for details see [7]). Let $\mathbb{C}=\{C_1,\ldots,C_m\}$ be a collection of subsets of $V$. Elements of $\mathbb{C}$ are called **cliques**. Let $T$ be a set of edges connecting pairs of cliques such that $(T,\mathbb{C})$ is a tree.

**Definition 1.** *Tree* $(T,\mathbb{C})$ *is a* **junction tree** *iff it satisfies the* **running intersection property (RIP)***:* $\forall C_i, C_j \in \mathbb{C}$ *and* $\forall C_k$ *on the (unique) simple path between* $C_i$ *and* $C_j$, $x \in C_i \cap C_j \Rightarrow x \in C_k$.

A set $S_{ij} \equiv C_i \cap C_j$ is called the **separator** corresponding to an edge $(i-j)$ from $T$. The size of a largest clique in a junction tree minus one is called the **treewidth** of that tree. For example, in a junction tree in Fig. 1, variable $x_2$ is contained in both clique 3 and 5, so it has to be contained in clique 2, because 2 is on the simple path between 3 and 5. The largest clique in Fig. 1 has size 3, so the treewidth of that junction tree is 2.

A distribution $P(V)$ is representable using junction tree $(T,\mathbb{C})$ if instantiating all variables in a separator $S_{ij}$ renders the variables on different sides of $S_{ij}$ independent. Denote the fact that $A$ is independent of $B$ given $C$ by $(A\perp B\,|\,C)$. Let $\mathbb{C}^i_{ij}$ be cliques that can be reached from $C_i$ in the $(T,\mathbb{C})$ without using edge $(i-j)$, and denote these reachable variables by $V^i_{ij} \equiv V^i_{ji} \equiv \bigcup_{C_k \in \mathbb{C}^i_{ij}} C_k \setminus S_{ij}$. For example, in Fig. 1, $S_{12}=\{x_1,x_5\}$, $V^1_{12}=\{x_4,x_6\}$, $V^2_{12}=\{x_2,x_3,x_7\}$.

**Definition 2.** $P(V)$ **factors according to junction tree** $(T,\mathbb{C})$ *iff* $\forall(i-j)\in T,\left(V^i_{ij}\perp V^j_{ij}\mid S_{ij}\right)$.

If a distribution $P(V)$ factors according to some junction tree of treewidth $k$, we will say that $P(V)$ is $k$**-JT representable**. In this case, a **projection** $P_{(T,\mathbb{C})}$ of $P$ on $(T,\mathbb{C})$, defined as

$$P_{(T,\mathbb{C})} = \frac{\prod_{C_i\in\mathbb{C}} P(C_i)}{\prod_{(i-j)\in T} P(S_{ij})}, \tag{1}$$

is equal to $P$ itself. For clarity, we will only consider **maximal** junction trees, where all separators have size $k$. If $P$ is $k$-JT representable, it also factors according to some maximal JT of treewidth $k$.

In practice the notion of conditional independence is too strong. Instead, a natural relaxation is to require sets of variables to have low **conditional mutual information** $I$. Denote $H(A)$ the entropy of $A$, then $I(A,B\,|\,S)\equiv H(A\,|\,S)-H(A\,|\,BS)$ is nonnegative, and zero iff $(A\perp B\,|\,S)$. Intuitively, $I\,(A,B\,|\,S)$ shows how much new information about $A$ can we extract from $B$ if we already know $S$.

**Definition 3.** $(T,\mathbb{C})$ *is an* $\varepsilon$**-junction tree** *for* $P(V)$ *iff* $\forall(i-j)\in T:\ I\left(V^i_{ij},V^j_{ij}\mid S_{ij}\right)\leq\varepsilon$.

If there exists an $\varepsilon$-junction tree $(T, \mathbb{C})$ for $P(V)$, we will say that $P$ is $k$-**JT** $\varepsilon$-**representable**. In this case, the Kullback-Leibler divergence of projection (1) of $P$ on $(T, \mathbb{C})$ from $P$ is bounded [13]:

$$KL\left(P, P_{(T,\mathbb{C})}\right) \leq n\varepsilon. \tag{2}$$

This bound means that if we have an $\varepsilon$-junction tree for $P(V)$, then instead of $P$ we can use its tractable principled approximation $P_{(T,\mathbb{C})}$ for inference. In this paper, we address the problem of **learning structure** of such junction tree from data (samples from $P$).

## 3 Structure learning

In this paper, we address the following problem: given data, such as multiple temperature readings from sensors in a sensor network, we treat each datapoint as an instantiation of the random variables $V$ and seek to find a good approximation of $P(V)$. We will assume that $P(V)$ is $k$-JT $\varepsilon$-representable for some $\varepsilon$ and aim to find a $\hat{\varepsilon}$-junction tree for $P$ with the same treewidth $k$ and with $\hat{\varepsilon}$ as small as possible. Note that the maximal treewidth $k$ is considered to be a constant and not a part of problem input. The complexity of our approach is exponential in $k$.

Let us initially assume that we have an oracle $I(\cdot, \cdot \mid \cdot)$ that can compute the mutual information $I(A, B \mid C)$ exactly for any disjoint subsets $A, B, C \subset V$. This is a very strict requirement, which we address in the next section. Using the oracle $I$, a naïve approach would be to evaluate[2] $I(Q, V_{-QS} \mid S)$ for all possible $Q, S \subset V$ s.t. $|S| = k$ and record all pairs $(S, Q)$ with $I(Q, V_{-QS} \mid S) \leq \varepsilon$ into a list $\mathbb{L}$. We will say that a junction tree $(T, \mathbb{C})$ is **consistent with** a **list** $\mathbb{L}$ iff for every separator $S_{ij}$ of $(T, \mathbb{C})$ it holds that $(S_{ij}, V_{ij}^i) \in \mathbb{L}$.

---

**Algorithm 2**: LTCI: find Conditional Independencies in Low-Treewidth distributions

**Input**: $V$, separator $S$, oracle $I(\cdot, \cdot \mid \cdot)$, threshold $\delta$, max set size $q$

1   $\mathbb{Q}_S \leftarrow \cup_{x \in V}\{x\}$ ;   // $\mathbb{Q}_S$ is a set of singletons

2   **for** $A \subset V_{-S}$ **s.t.** $|A| \leq q$ **do**

3     **if** $\min_{X \subset A} I(X, A_{-X} \mid S) > \delta$ **then**

      // find min with Queyranne's alg.

4       merge all $Q_i \in \mathbb{Q}_S$, s.t. $Q_i \cap A \neq \emptyset$

5   **return** $\mathbb{Q}_S$

---

After $\mathbb{L}$ is formed, any junction tree consistent with $\mathbb{L}$ would be an $\varepsilon$-junction tree for $P(V)$. Such tree would be found by some *FindConsistentTree* procedure, implemented, e.g., using constraint satisfaction. Alg. 1 summarizes this idea. Algorithms that follow this outline, including ours, form a class of *constraint-based* approaches. These algorithms use mutual information tests to constrain the set of possible structures and return one that is consistent with the constraints. Unfortunately, using Alg. 1 directly is impractical because its complexity is exponential in the total number of variables $n$. In the following sections we discuss inefficiencies of Alg. 1 and present efficient solutions.

### 3.1 Global independence assertions from local tests

One can see two problems with the inner loop of Alg. 1 (lines 3-5). First, for each separator we need to call the oracle exponentially many times ($2^{n-k-1}$, once for every $Q \subset V_{-S}$). This drawback is addressed in the next section. Second, the mutual information oracle, $I(A, B \mid S)$, is called on subsets $A$ and $B$ of size $O(n)$. Unfortunately, the best known way of computing mutual information (and estimating $I$ from data) has time and sample complexity exponential in $|A| + |B| + |S|$. Previous work has not addressed this problem. In particular, the approach of [13] has exponential complexity, in general, because it needs to estimate $I$ for subsets of size $O(n)$. Our first new result states that we can limit ourselves to computing mutual information over small subsets of variables:

**Lemma 4.** *Let $P(V)$ be a $k$-JT $\varepsilon$-representable distribution. Let $S \subset V$, $A \subset V_{-S}$. If $\forall X \subseteq V_{-S}$ s.t. $|X| \leq k+1$, it holds that $I(A \cap X, V_{-SA} \cap X \mid S) \leq \delta$, then $I(A, V_{-SA} \mid S) \leq n(\varepsilon + \delta)$.*

We can thus compute an upper bound on $I(A, V_{-SA} \mid S)$ using $O\left(\binom{n}{k}\right) \equiv O(n^k)$ (i.e., polynomially many) calls to the oracle $I(\cdot, \cdot \mid \cdot)$, and each call will involve at most $|S| + k + 1$ variables. Lemma 4 also bounds the quality of approximation of $P$ by a projection on any junction tree $(T, \mathbb{C})$:

**Corollary 5.** *If conditions of Lemma 4 hold for $P(V)$ with $S = S_{ij}$ and $A = V_{ij}^i$ for every separator $S_{ij}$ of a junction tree $(T, \mathbb{C})$, then $(T, \mathbb{C})$ is a $n(\varepsilon + \delta)$-junction tree for $P(V)$.*

### 3.2 Partitioning algorithm for weak conditional independencies

Now that we have an efficient upper bound for $I(\cdot, \cdot \mid \cdot)$ oracle, let us turn to reducing the number of oracle calls by Alg. 1 from exponential ($2^{n-k-1}$) to polynomial. In [13], Narasimhan and Bilmes

| **Algorithm 3**: Efficient approach to structure learning | **Algorithm 4**: FindConsistentTreeDPGreedy |
|---|---|
| **Input**: $V$, oracle $I\left(\cdot,\cdot\mid\cdot\right)$, treewidth $k$, threshold $\varepsilon$, $\mathbb{L}=\emptyset$ | **Input**: List $\mathbb{L}$ of components $(S,Q)$ |
| **1 for** $S\subset V$ *s.t.* $\|S\|=k$ **do** | **1 for** $(S,Q)\in\mathbb{L}$ *in the order of increasing* $\|Q\|$ **do** |
| **2**    **for** $Q\in$ *LTCI(V,S,I,$\varepsilon$,k + 2)* **do** | **2**    greedily check if $(S,Q)$ is $\mathbb{L}$-decomposable |
| **3**      $\mathbb{L}\leftarrow\mathbb{L}\cup(S,Q)$ | **3**    record the decomposition if it exists |
| | **4 if** $\exists S:(S,V_{-S})$ *is* $\mathbb{L}$-*decomposable* **then** |
| **4 return** *FindConsistentTreeDPGreedy($\mathbb{L}$)* | **5**    **return** corresponding junction tree |
| | **6 else return** no tree found |

present an approximate solution to this problem, assuming that an efficient approximation of oracle $I\left(\cdot,\cdot\mid\cdot\right)$ exists. A key observation that they relied on is that the function $F_S(A)\equiv I\left(A,V_{-SA}\mid S\right)$ is **submodular**: $F_S(A)+F_S(B)\geq F_S(A\cup B)+F_S(A\cap B)$. Queyranne's algorithm [14] allows the minimization of a submodular function $F$ using $O(n^3)$ evaluations of $F$. [13] combines Queyranne's algorithm with divide-and-conquer approach to partition $V_{-S}$ into conditionally independent subsets using $O(n^3)$ evaluations of $I\left(\cdot,\cdot\mid\cdot\right)$. However, since $I\left(\cdot,\cdot\mid\cdot\right)$ is computed for sets of size $O(n)$, complexity of their approach is still exponential in $n$, in general.

Our approach, called LTCI (Alg. 2), in contrast, has polynomial complexity for $q=O(1)$. We will show that $q=O(1)$ in our approach that uses LTCI as a subroutine. To gain intuition for LTCI, suppose there exists a $\varepsilon$-junction tree for $P(V)$, such that $S$ is a separator and subsets $B$ and $C$ are on different sides of $S$ in the junction tree. By definition, this means $I\left(B,C\mid S\right)\leq\varepsilon$. When we look at subset $A\equiv B\cup C$, the true partitioning is not known, but setting $\delta=\varepsilon$, we can test all possible $2^{\|A\|-1}$ ways to partition $A$ into two subsets ($X$ and $A_{-X}$). If none of the possible partitionings have $I\left(X,A_{-X}\mid S\right)\leq\varepsilon$, we can conclude that all variables in $A$ are on the same side of separator $S$ in any $\varepsilon$-junction tree that includes $S$ as a separator. Notice also that

$$\forall X\subset A\ \ I\left(X,A_{-X}\mid S\right)>\delta\Leftrightarrow\min_{X\subset A}I\left(X,A_{-X}\mid S\right)>\delta,$$

so we can use Queyranne's algorithm to evaluate $I\left(\cdot,\cdot\mid\cdot\right)$ only $O(\|A\|^3)$ times instead of $2^{\|A\|-1}$ times for minimization by exhaustive search. LTCI initially assumes that every variable $x$ forms its own partition $Q=\{x\}$. If a test shows that two variables $x$ and $y$ are on the same side of the separator, it follows that their container partitions $Q_1\ni x,Q_2\ni y$ cannot be separated by $S$, so LTCI merges $Q_1$ and $Q_2$ (line 3 of Alg. 2). This process is then repeated for larger sets of variables, of size up to $q$, until we converge to a set of partitions that are "almost independent" given $S$.

**Proposition 6.** *The time complexity of LTCI with* $\|S\|=k$ *is* $O\left(\binom{n}{q}nJ_{k+q}^{MI}\right)\equiv O\left(n^{q+1}J_{k+q}^{MI}\right)$, *where* $J_{k+q}^{MI}$ *is the time complexity of computing* $I\left(A,B\mid C\right)$ *for* $\|A\|+\|B\|+\|C\|=k+q$.

It is important that the partitioning algorithm returns partitions that are similar to connected components of $V_{ij}^i$ of the true junction tree for $P(V)$. Formally, let us define two desirable properties. Suppose $(T,\mathbb{C})$ is an $\varepsilon$-junction tree for $P(V)$, and $\mathbb{Q}_{S_{ij}}$ is an output of the algorithm for separator $S_{ij}$ and threshold $\delta$. We will say that partitioning algorithm is **correct** iff for $\delta=\varepsilon$, $\forall Q\in\mathbb{Q}_{S_{ij}}$ either $Q\subseteq V_{ij}^i$ or $Q\subseteq V_{ij}^j$. A correct algorithm will never mistakenly put two variables on the same side of a separator. We will say that an algorithm is $\alpha$-**weak** iff $\forall Q\in\mathbb{Q}_{S_{ij}}I\left(Q,V_{-QS_{ij}}\mid S_{ij}\right)\leq\alpha$. For small $\alpha$, an $\alpha$-weak algorithm puts variables on different sides of a separator only if corresponding mutual information between those variables is not too large. Ideally, we want a correct and $\delta$-weak algorithm; for $\delta=\varepsilon$ it would separate variables that are on different sides of $S$ in a true junction tree, but not introduce any spurious independencies. LTCI, which we use instead of lines 3-5 in Alg. 1, satisfies the first requirement and a relaxed version of the second:

**Lemma 7.** *LTCI, for* $q\geq k+1$, *is correct and* $n(\varepsilon+(k-1)\delta)$-*weak*.

### 3.3 Implementing *FindConsistentTree* using dynamic programming

A concrete form of *FindConsistentTree* procedure is the last step needed to make Alg. 1 practical. For *FindConsistentTree*, we adopt a dynamic programming approach from [2] that was also used in [13] for the same purpose. We briefly review the intuition; see [2] for details.

Consider a junction tree $(T,\mathbb{C})$. Let $S_{ij}$ be a separator in $(T,\mathbb{C})$ and $\mathbb{C}_{ij}^i$ be the set of cliques reachable from $C_i$ without using edge $(i-j)$. Denote $T_{ij}^i$ the set of edges from $T$ that connect

cliques from $\mathbb{C}_{ij}^i$. If $(T, \mathbb{C})$ is an $\varepsilon$-junction tree for $P(V)$, then $(\mathbb{C}_{ij}^i, T_{ij}^i)$ is an $\varepsilon$-junction tree for $P(V_{ij}^i \cup S_{ij})$. Moreover, the subtree $(\mathbb{C}_{ij}^i, T_{ij}^i)$ consists of a clique $C_i$ and several sub-subtrees that are each connected to $C_i$. For example, in Fig. 1 the subtree over cliques 1,2,4,5 can be decomposed into clique 2 and two sub-subtrees: one including cliques $\{1,4\}$ and one with clique 5. The recursive structure suggests dynamic programming approach: given a component $(S, Q)$ such that $I(Q, V_{-QS} \mid S) < \delta$, check if smaller subtrees can be put together to cover the variables of $(S, Q)$. Formally, we require the following property:

**Definition 8.** $(S, Q) \in \mathbb{L}$ *is* $\mathbb{L}$-**decomposable** *iff* $\exists \mathbb{D} = \cup_i \{(S_i, Q_i)\}, x \in Q$ *s.t.*

    *1.* $\forall i (S_i, Q_i)$ *is* $\mathbb{L}$-**decomposable** *and* $\cup_{i=1}^m Q_i = Q \setminus \{x\}$;

    *2.* $S_i \subset S \cup \{x\}$, *i.e., each subcomponent can be connected directly to the clique* $(S, x)$;

    *3.* $Q_i \cap Q_j = \emptyset$, *ensuring the running intersection property within the subtree over* $S \cup Q$.

*The set* $\{(S_1, Q_1), \dots, (S_m, Q_m)\}$ *is called a* **decomposition** *of* $(S, Q)$.

Unfortunately, checking whether a decomposition exists is equivalent to an NP-complete *exact set cover* problem because of the requirement $Q_i \cap Q_j = \emptyset$ in part 3 of Def. 8. Unfortunately, this challenging issue was not addressed by [13], where the same algorithm was used. To keep complexity polynomial, we use a simple greedy approach: for every $x \in Q_i$, starting with an empty candidate decomposition $\mathbb{D}$, add $(S_i, Q_i) \in \mathbb{L}$ to $\mathbb{D}$ if the last two properties of Def. 8 hold for $(S_i, Q_i)$. If eventually Def. 8 holds, return the decomposition $\mathbb{D}$, otherwise return that no decomposition exists. We call the resulting procedure *FindConsistentTreeDPGreedy*.

**Proposition 9.** *For separator size $k$, time complexity of FindConsistentTreeDPGreedy is $O(n^{k+2})$*

Combining Alg. 2 and *FindConsistentTreeDPGreedy*, we arrive at Alg. 3. Overall complexity of Alg. 3 is dominated by Alg. 2 and is equal to $O(n^{2k+3} J_{2k+2}^{MI})$.

In general, *FindConsistentTreeDP* with greedy decomposition checks may miss a junction tree that is consistent with the list of components $\mathbb{L}$, but there is a class of distributions for which Alg. 3 is guaranteed to find a junction tree. Intuitively, we require that for every $(S_{ij}, V_{ij}^i)$ from a $\varepsilon$-junction tree $(T, \mathbb{C})$, Alg. 2 adds all the components from decomposition of $(S_{ij}, V_{ij}^i)$ to $\mathbb{L}$ and nothing else. This requirement is guaranteed for distributions where variables inside every clique of the junction tree are sufficiently strongly interdependent (have a certain level of mutual information):

**Lemma 10.** *If $\exists$ an $\varepsilon$-JT $(T, \mathbb{C})$ for $P(V)$ s.t. no two edges of $T$ have the same separator, and for every separator $S$, clique $C \in \mathbb{C}$, $\min_{X \subset C_{-S}} I(X, C_{-XS} \mid S) > (k+3)\varepsilon$ (we will call $(T, \mathbb{C})$ $(k+3)\varepsilon$-**strongly connected**), then Alg. 3, called with $\delta = \varepsilon$, will output a $nk\varepsilon$-JT for $P(V)$.*

## 4 Sample complexity

So far we have assumed that a mutual information oracle $I(\cdot, \cdot \mid \cdot)$ exists for the distribution $P(V)$ and can be efficiently queried. In real life, however, one only has data (i.e., samples from $P(V)$) to work with. However, we can get a probabilistic estimate of $I(A, B \mid C)$, that has accuracy $\pm\Delta$ with probability $1 - \gamma$, using number of samples and computation time polynomial in $\frac{1}{\Delta}$ and $\log \frac{1}{\gamma}$:

**Theorem 11.** *(Höffgen, [9]). The entropy of a probability distribution over $2k + 2$ discrete variables with domain size $R$ can be estimated with accuracy $\Delta$ with probability at least $(1 - \gamma)$ using $F(k, R, \Delta, \gamma) \equiv O\left(\frac{R^{4k+4}}{\Delta^2} \log^2\left(\frac{R^{2k+2}}{\Delta^2}\right) \log\left(\frac{R^{2k+2}}{\gamma}\right)\right)$ samples from $P$ and the same amount of time.*

If we employ this oracle in our algorithms, the performance guarantee becomes probabilistic:

**Theorem 12.** *If there exists a $(k+3)(\varepsilon + 2\Delta)$-strongly connected $\varepsilon$-junction tree for $P(V)$, then Alg. 3, called with $\delta = \varepsilon + \Delta$ and $\hat{I}(\cdot, \cdot, \cdot)$ based on Thm. 11, using $U \equiv F(k, R, \Delta, \frac{\gamma}{n^{2k+2}})$ samples and $O(n^{2k+3}U)$ time, will find a $kn(\varepsilon + 2\Delta)$-junction tree for $P(V)$ with probability at least $(1 - \gamma)$.*

Finally, if $P(V)$ is $k$-JT representable (i.e., $\varepsilon = 0$), and the corresponding junction tree is strongly connected, then we can let both $\Delta$ and $\gamma$ go to zero and use Alg. 3 to find, with probability arbitrarily close to one, a junction tree that approximates $P$ arbitrarily well in time polynomial in $\frac{1}{\Delta}$ and $\log \frac{1}{\gamma}$, i.e., the class of strongly connected $k$-junction trees is probably approximately correctly learnable[3].

**Corollary 13.** *If there exists an $\alpha$-strongly connected junction tree for $P(V)$ with $\alpha > 0$, then for $\beta < \alpha n$, Alg. 3 will learn a $\beta$-junction tree for $P$ with probability at least $1 - \gamma$ using $O\left(\frac{n^4}{\beta^2} \log^2 \frac{n}{\beta} \log \frac{n}{\gamma}\right)$ samples from $P(V)$ and $O\left(\frac{n^{2k+7}}{\beta^2} \log^2 \frac{n}{\beta} \log \frac{n}{\gamma}\right)$ computation time.*

## 5  Lazy evaluation of mutual information

Alg. 3 requires the value of threshold $\delta$ as an input. To get tighter quality guarantees, we need to choose the smallest $\delta$ for which Alg. 3 finds a junction tree. *A priori*, this value is not known, so we need a procedure to choose the optimal $\delta$. A natural way to select $\delta$ is binary search. For discrete random variables with domain size $R$, for any $P(V), S, x$ it holds that $I\left(x, V_{-Sx} \mid S\right) \leq \log R$, so for any $\delta > \log R$ Alg. 3 is guaranteed to find a junction tree (with all cliques connected to the same separator). Thus, we can restrict binary search to range $\delta \in [0, \log R]$.

In binary search, for every value of $\delta$, Alg. 2 checks the result of Queyranne's algorithm minimizing $\min_{X \subset A} I\left(X, A_{-X} \mid S\right)$ for every $|S| = k, |A| \leq k+2$, which amounts to $O(n^{2k+2})$ complexity per value of $\delta$. It is possible, however, to find the optimal $\delta$ while only checking $\min_{X \subset A} I\left(X, A_{-X} \mid S\right)$ for every $S$ and $A$ once over the course of the search process. Intuitively, think of the set of partitions $\mathbb{Q}_S$ in Alg. 2 as a set of connected components of a graph with variables as vertices, and a hyper-edge connecting all variables from $A$ whenever $\min_{X \subset A} I\left(X, A_{-X} \mid S\right) > \delta$. As $\delta$ increases, some of the hyper-edges disappear, and the number of connected components (or independent sets) may increase. More specifically, a graph $\mathbb{Q}_S$ is maintained for each separator $S$. For all $S, A$ add a hyper-edge connecting all variables in $A$ annotated with $\text{strength}_S(A) \equiv \min_{X \subset A} I\left(X, A_{-X} \mid S\right)$ to $\mathbb{Q}_S$. Until $FindConsistentTree(\cup_S \mathbb{Q}_S)$ returns a tree, increase $\delta$ to be $\min_{S, A: \text{hyperedge}_S(A) \in \mathbb{Q}_S} \text{strength}_S(A)$ (i.e., strength of the weakest remaining hyper-edge), and remove $\text{hyperedge}_S(A)$ from $\mathbb{Q}_S$. Fig. 2(a) shows an example evolution of $\mathbb{Q}_{x_4}$ for $k = 1$.

To further save computation time, we exploit two observations: First, if $A$ is a subset of a connected component $Q \in \mathbb{Q}_S$, adding $\text{hyperedge}_S(A)$ to $\mathbb{Q}_S$ will not change $\mathbb{Q}_S$. Thus, we do not test any hyper-edge $A$ which is contained in a connected component. However, as $\delta$ increases, a component may become disconnected, because such an edge was not added. Therefore, we may have more components than we should (inducing incorrect independencies). This issue is addressed by our second insight: If we find a junction tree for a particular value of $\delta$, we only need to recheck the components used in this tree. These insights lead to a simple, *lazy* procedure: If *FindConsistentTree* returns a tree $(T, \mathbb{C})$, we check the hyper-edges that intersect the components used to form $(T, \mathbb{C})$. If none of these edges are added, then we can return $(T, \mathbb{C})$ for this value of $\delta$. Otherwise, some of $\mathbb{Q}_S$ have changed; we can iterate this procedure until we find a solution.

## 6  Evaluation

To evaluate our approach, we have applied it to two real-world (sensor network temperature [8] and San Francisco Bay area traffic [11]) and one artificial (samples from ALARM Bayesian network [4]) datasets. Our implementation, called LPACJT, uses lazy evaluations of $I\left(\cdot, \cdot \mid \cdot\right)$ from section 5. As baselines for comparison, we used a simple hill-climbing heuristic[4], a combination of LPACJT with hill-climbing, where intermediate results returned by *FindConsistentTree* were used as starting points for hill-climbing, Chow-Liu algorithm, and algorithms of [10] (denoted Karger-Srebro) and [17] (denoted OBS). All experiments were run on a Pentium D 3.4 GHz, with runtimes capped to 10 hours. The necessary entropies were cached in advance.

**ALARM**. This discrete-valued data was sampled from a known Bayesian network with treewidth 4. We learned models with treewidth 3 because of computational concerns. Fig. 2(b) shows the per-point log-likelihood of learned models on test data depending on the amount of training data. We see that on small training datasets both LPACJT finds better models than a basic hill-climbing approach, but worse than the OBS of [17] and Chow-Liu. The implementation of OBS was the only one to use regularization, so this outcome can be expected. We can also conclude that on this dataset our approach overfits than hill-climbing. For large enough training sets, LPACJT results achieve the likelihood of the true model, despite being limited to models with smaller treewidth. Chow-Liu performs much worse, since it is limited to models with treewidth 1. Fig. 2(c) shows an example of a structure found by LPACJT for ALARM data. LPACJT only missed 3 edges of the true model.

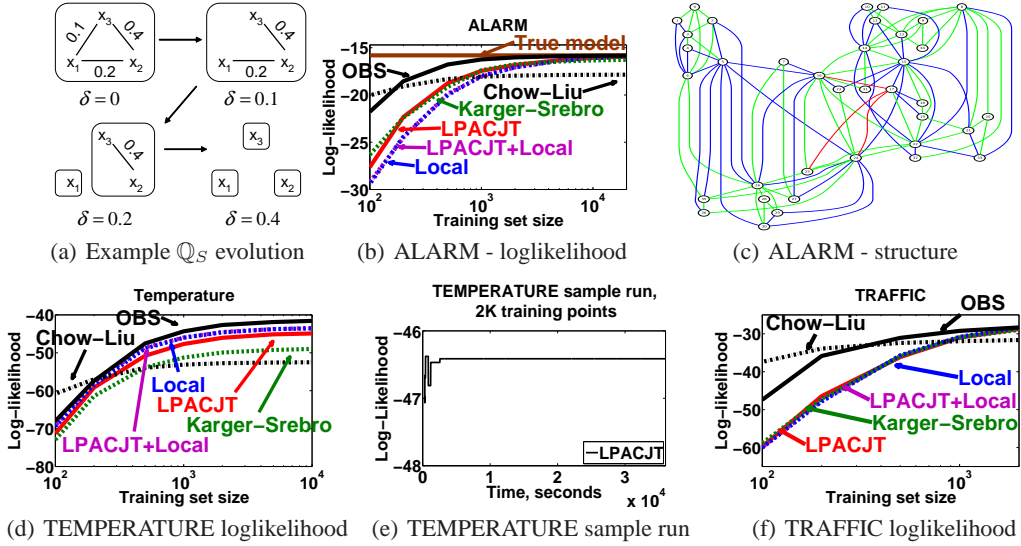

| | | |
|---|---|---|
| (a) Example $\mathbb{Q}_S$ evolution | (b) ALARM - loglikelihood | (c) ALARM - structure |
| (d) TEMPERATURE loglikelihood | (e) TEMPERATURE sample run | (f) TRAFFIC loglikelihood |

Figure 2: An example of evolution of $\mathbb{Q}_S$ for section 5 (2(a)), one structure learned by LPACJT(2(c)), experimental results (2(b),2(d),2(f)), and an example evolution of the test set likelihood of the best found model (2(e)). In 2(c), nodes denote variables, edges connect variables that belong to the same clique, green edges belong to both true and learned models, blue edges belong only to the learned model, red - only to the true one.

**TEMPERATURE**. This data is from a 2-month deployment of 54 sensor nodes (15K datapoints) [8]. Each variable was discretized into 4 bins and we learned models of treewidth 2. Since the locations of the sensor have an $\infty$-like shape with two loops, the problem of learning a thin junction tree for this data is hard. In Fig. 2(d) one can see that LPACJT performs almost as good as hill-climbing-based approaches, and, on large training sets, much better than Karger-Srebro algorithm. Again, as expected, LPACJT outperforms Chow-Liu algorithm by a significant margin if there is enough data available, but overfits on the smallest training sets. Fig 2(e) shows the evolution of the test set likelihood of the best (highest training set likelihood) structure identified by LPACJT over time. The first structure was identified within 5 minutes, and the final result within 1 hour.

**TRAFFIC**. This dataset contains traffic flow information measured every 5 minutes in 8K locations in California for 1 month [11]. We selected 32 locations in San Francisco Bay area for the experiments, discretized traffic flow values into 4 bins and learned models of treewidth 3. All non-regularized algorithms, including LPACJT, give results of essentially the same quality.

## 7 Relation to prior work and conclusions

For a brief overview of the prior work, we refer the reader to Fig. 3. Most closely related to LPACJT are learning factor graphs of [1] and learning limited-treewidth Markov nets of [13, 10]. Unlike our approach, [1] does not guarantee low treewidth of the result, instead settling for compactness. [13, 10] guarantee low treewidth. However, [10] only guarantees that the difference of the log-likelihood of the result from the fully independent model is within a constant factor from the difference of the most likely JT: $LLH(\text{optimal}) - LLH(\text{indep.}) \leq 8^k k!^2 (LLH(\text{learned}) - LLH(\text{indep.}))$. [13] has exponential complexity. Our approach has polynomial complexity and quality guarantees that hold for strongly connected $k$-JT $\varepsilon$-representable distributions, while those of [13] only hold for $\varepsilon = 0$.

We have presented the first truly polynomial algorithm for learning junction trees with limited treewidth. Based on a new upper bound for conditional mutual information that can be computed using polynomial time and number of samples, our algorithm is guaranteed to find a junction tree that is close in $KL$ divergence to the true distribution, for *strongly connected* $k$-JT $\varepsilon$-representable distributions. As a special case of these guarantees, we show PAC-learnability of strongly connected $k$-JT representable distributions. We believe that the new theoretical insights herein provide significant step in the understanding of structure learning in graphical models, and are useful for the analysis of other approaches to the problem. In addition to the theory, we have also demonstrated experimentally that these theoretical ideas are viable, and can, in the future, be used in the development of fast and effective structure learning heuristics.

| approach | model class | guarantees | true distribution | samples | time | reference |
|---|---|---|---|---|---|---|
| score | tractable | local | any | any | poly$^\dagger$ | [3, 5] |
| score | tree | global | any | any | $O(n^2)$ | [6] |
| score | tree mixture | local | any | any | $O(n^2)^\dagger$ | [12] |
| score | compact | local | any | any | poly$^\dagger$ | [17] |
| score | all | global | any | any | exp | [15] |
| score | tractable | const-factor | any | any | poly | [10] |
| constraint | compact | PAC$^\circ$ | positive | poly | poly | [1] |
| constraint | all | global | any | $\infty$ | poly(tests) | [16] |
| constraint | tractable | PAC | strong $k$-JT | exp$^\ddagger$ | exp$^\ddagger$ | [13] |
| constraint | tractable | PAC$^\S$ | strong $k$-JT | poly | poly | this paper |

Figure 3: Prior work. The majority of the literature can be subdivided into score-based [3, 5, 6, 12, 15, 10] and constraint-based [13, 16, 1] approaches. The former try to maximize some target function, usually regularized likelihood, while the latter perform conditional independence tests and restrict the set of candidate structures to those consistent with the results of the tests. *Tractable* means that the result is guaranteed to be of limited treewidth, *compact* - with limited connectivity of the graph. *Guarantees* column shows whether the result is a local or global optimum, whether there are PAC guarantees, or whether the difference of the log-likelihood of the result from the fully independent model is within a *const-factor* from the difference of the most likely JT. *True distribution* shows for what class of distributions the guarantees hold. † superscript means per-iteration complexity, poly - $O(n^{O(k)})$, exp$^\ddagger$ - exponential in general, but poly for special cases. PAC$^\circ$ and PAC$^\S$ mean PAC with (different) graceful degradation guarantees.

# 8 Acknowledgments

This work is supported in part by NSF grant IIS-0644225 and by the ONR under MURI N000140710747. C. Guestrin was also supported in part by an Alfred P. Sloan Fellowship. We thank Nathan Srebro for helpful discussions, and Josep Roure, Ajit Singh, CMU AUTON lab, Mark Teyssier, Daphne Koller, Percy Liang and Nathan Srebro for sharing their source code.

## Footnotes

[1]Notation note: throughout the paper, we use small letters $(x,y)$ to denote variables, capital letters $(V,C)$ to denote sets of variables, and double-barred font $(\mathbb{C},\mathbb{D})$ to denote sets of sets.

[2]Notation note: for any sets $A, B, C$ we will denote $A \setminus (B \cup C)$ as $A_{-BC}$ to lighten the notation.

[3]A class $\mathbb{P}$ of distributions is PAC learnable if for any $P \in \mathbb{P}, \delta > 0, \gamma > 0$ a learning algorithm will output $P' : KL(P, P') < \delta$ with probability $1 - \gamma$ in time polynomial in $\frac{1}{\delta}$ and $\log \frac{1}{\gamma}$.

[4]Hill-climbing had 2 kinds of moves available: replace variable $x$ with variable $y$ in a connected subjunction tree, or relpace a leaf clique $C_i$ with another clique $(C_i \setminus S_{ij}) \cup S_{mr}$ connected to a separator $S_{mr}$.

# References

[1] P. Abbeel, D. Koller, and A. Y. Ng. Learning factor graphs in polynomial time and sample complexity. *JMLR*, 7, 2006.

[2] S. Arnborg, D. G. Corneil, and A. Proskurowski. Complexity of finding embeddings in a k-tree. *SIAM Journal on Algebraic and Discrete Methods*, 8(2):277–284, 1987.

[3] F. R. Bach and M. I. Jordan. Thin junction trees. In *NIPS*, 2002.

[4] I. Beinlich, J. Suermondt, M. Chavez, and G. Cooper. The ALARM monitoring system: A case study with two probablistic inference techniques for belief networks. In *Euro. Conf. on AI in Medicine*, 1988.

[5] A. Choi, H. Chan, and A. Darwiche. On Bayesian network approximation by edge deletion. In *UAI*, 2005.

[6] C. Chow and C. Liu. Approximating discrete probability distributions with dependence trees. *IEEE Transactions on Information Theory*, 14(3):462–467, 1968.

[7] R. G. Cowell, P. A. Dawid, S. L. Lauritzen, and D. J. Spiegelhalter. *Probabilistic Networks and Expert Systems (Information Science and Statistics)*. Springer, May 2003.

[8] A. Deshpande, C. Guestrin, S. Madden, J. Hellerstein, and W. Hong. Model-driven data acquisition in sensor networks. In *VLDB*, 2004.

[9] K. U. Höffgen. Learning and robust learning of product distributions. In *COLT*, 1993.

[10] D. Karger and N. Srebro. Learning Markov networks: Maximum bounded tree-width graphs. SODA-01.

[11] A. Krause and C. Guestrin. Near-optimal nonmyopic value of information in graphical models. UAI-05.

[12] M. Meilă and M. I. Jordan. Learning with mixtures of trees. *JMLR*, 1:1–48, 2001.

[13] M. Narasimhan and J. Bilmes. PAC-learning bounded tree-width graphical models. In *UAI*, 2004.

[14] M. Queyranne. Minimizing symmetric submodular functions. *Math. Programming*, 82(1):3–12, 1998.

[15] A. Singh and A. Moore. Finding optimal Bayesian networks by dynamic programming. Technical Report CMU-CALD-05-106, Carnegie Mellon University, Center for Automated Learning and Discovery, 2005.

[16] P. Spirtes, C. Glymour, and R. Scheines. *Causation, Prediction, and Search*. MIT Press, 2001.

[17] M. Teyssier and D. Koller. Ordering-based search: A simple and effective algorithm for learning Bayesian networks. In *UAI*, 2005.

